# Locally Adaptive Nearest Neighbor Algorithms

**Dietrich Wettschereck**        **Thomas G. Dietterich**
Department of Computer Science
Oregon State University
Corvallis, OR 97331-3202
`wettscd@cs.orst.edu`

## Abstract

Four versions of a $k$-nearest neighbor algorithm with locally adaptive $k$ are introduced and compared to the basic $k$-nearest neighbor algorithm (kNN). Locally adaptive kNN algorithms choose the value of $k$ that should be used to classify a query by consulting the results of cross-validation computations in the local neighborhood of the query. Local kNN methods are shown to perform similar to kNN in experiments with twelve commonly used data sets. Encouraging results in three constructed tasks show that local methods can significantly outperform kNN in specific applications. Local methods can be recommended for on-line learning and for applications where different regions of the input space are covered by patterns solving different sub-tasks.

## 1 Introduction

The k-nearest neighbor algorithm (kNN, Dasarathy, 1991) is one of the most venerable algorithms in machine learning. The entire training set is stored in memory. A new example is classified with the class of the majority of the $k$ nearest neighbors among all stored training examples. The (global) value of $k$ is generally determined via cross-validation.

For certain applications, it might be desirable to vary the value of $k$ locally within different parts of the input space to account for varying characteristics of the data such as noise or irrelevant features. However, for lack of an algorithm, researchers have assumed a global value for $k$ in all work concerning nearest neighbor algorithms to date (see, for example, Bottou, 1992, p. 895, last two paragraphs of Section 4.1). In this paper, we propose and evaluate four new algorithms that determine different values for $k$ in different parts of the input space and apply these varying values to classify novel examples. These four algorithms use different methods to compute the $k$-values that are used for classification.

We determined two basic approaches to compute locally varying values for $k$. One could compute a single $k$ or a set of $k$ values for each training pattern, or training patterns could be combined into groups and $k$ value(s) computed for these groups. A procedure to determine the $k$ to be used at classification time must be given in both approaches. Representatives of these two approaches are evaluated in this paper and compared to the global kNN algorithm. While it was possible to construct data sets where local algorithms outperformed kNN, experiments with commonly used data sets showed, in most cases, no significant differences in performance. A possible explanation for this behavior is that data sets which are commonly used to evaluate machine learning algorithms may all be similar in that attributes such as distribution of noise or irrelevant features are uniformly distributed across all patterns. In other words, patterns from data sets describing a certain task generally exhibit similar properties.

Local nearest neighbor methods are comparable in computational complexity and accuracy to the (global) k-nearest neighbor algorithm and are easy to implement. In specific applications they can significantly outperform kNN. These applications may be combinations of significantly different subsets of data or may be obtained from physical measurements where the accuracy of measurements depends on the value measured. Furthermore, local kNN classifiers can be constructed at classification time (on-line learning) thereby eliminating the need for a global cross-validation run to determine the proper value of $k$.

## 1.1 Methods compared

The following nearest neighbor methods were chosen as representatives of the possible nearest neighbor methods discussed above and compared in the subsequent experiments:

- k-nearest neighbor (kNN)

   This algorithm stores all of the training examples. A single value for $k$ is determined from the training data. Queries are classified according to the class of the majority of their $k$ nearest neighbors in the training data.

- localKNN$_{ks\ unrestricted}$

   This is the basic local kNN algorithm. The three subsequent algorithms are modifications of this method. This algorithm also stores all of the training examples. Along with each training example, it stores a list of those values of $k$ that correctly classify that example under leave-one-out cross-validation. To classify a query $q$, the $M$ nearest neighbors of the query are computed, and that $k$ which classifies correctly most of these $M$

neighbors is determined. Call this value $k_{M,q}$. The query $q$ is then classified with the class of the majority of its $k_{M,q}$ nearest neighbors. Note that $k_{M,q}$ can be larger or smaller than $M$. The parameter $M$ is the only parameter of the algorithm, and it can be determined by cross-validation.

- localKNN$_{ks\ pruned}$
  The list of $k$ values for each training example generally contains many values. A global histogram of $k$ values is computed, and $k$ values that appear fewer than $L$ times are pruned from all lists (at least one $k$ value must, however, remain in each list). The parameter $L$ can be estimated via cross-validation. Classification of queries is identical to localKNN$_{ks\ unrestricted}$.

- localKNN$_{one\ k\ per\ class}$
  For each output class, the value of $k$ that would result in the correct (leave-one-out) classification of the maximum number of training patterns from that class is determined. A query $q$ is classified as follows: Assume there are two output classes, $C_1$ and $C_2$. Let $k_1$ and $k_2$ be the $k$ value computed for classes $C_1$ and $C_2$, respectively. The query is assigned to class $C_1$ if the percentage of the $k_1$ nearest neighbors of $q$ that belong to class $C_1$ is larger than the percentage of the $k_2$ nearest neighbors of $q$ that belong to class $C_2$. Otherwise, $q$ is assigned to class $C_2$. Generalization of that procedure to any number of output classes is straightforward.

- localKNN$_{one\ k\ per\ cluster}$
  An unsupervised cluster algorithm (RPCL,[1] Xu et al., 1993) is used to determine clusters of input data. A single $k$ value is determined for each cluster. Each query is classified according to the $k$ value of the cluster it is assigned to.

## 2  Experimental Methods and Data sets used

To measure the performance of the different nearest neighbor algorithms, we employed the training set/test set methodology. Each data set was randomly partitioned into a training set containing approximately 70% of the patterns and a test set containing the remaining patterns. After training on the training set, the percentage of correct classifications on the test set was measured. The procedure was repeated a total of 25 times to reduce statistical variation. In each experiment, the algorithms being compared were trained (and tested) on identical data sets to ensure that differences in performance were due entirely to the algorithms. Leave-one-out cross-validation (Weiss & Kulikowski, 1991) was employed in all experiments to estimate optimal settings for free parameters such as $k$ in kNN and $M$ in localKNN.

We report the average percentage of correct classifications and its standard error. Two-tailed paired t-tests were conducted to determine at what level of significance one algorithm outperforms the other. We state that one algorithm significantly outperforms another when the p-value is smaller than 0.05.

# 3   Results

## 3.1   Experiments with Constructed Data Sets

Three experiments with constructed data sets were conducted to determine the ability of local nearest neighbor methods to determine proper values of $k$. The data sets were constructed such that it was known before experimentation that varying $k$ values should lead to superior performance. Two data sets which were presumed to require significantly different values of $k$ were combined into a single data set for each of the first two experiments. For the third experiment, a data set was constructed to display some characteristics of data sets for which we assume local kNN methods would work best. The data set was constructed such that patterns from two classes were stretched out along two parallel lines in one part of the input space. The parallel lines were spaced such that the nearest neighbor for most patterns belongs to the same class as the pattern itself, while two out of the three nearest neighbors belong to the other class. In other parts of the input space, classes were well separated, but class labels were flipped such that the nearest neighbor of a query may indicate the wrong pattern while the majority of the $k$ nearest neighbors ($k > 3$) would indicate the correct class (see also Figure 4).

Figure 1 shows that in selected applications, local nearest neighbor methods can lead to significant improvements over kNN in predictive accuracy.

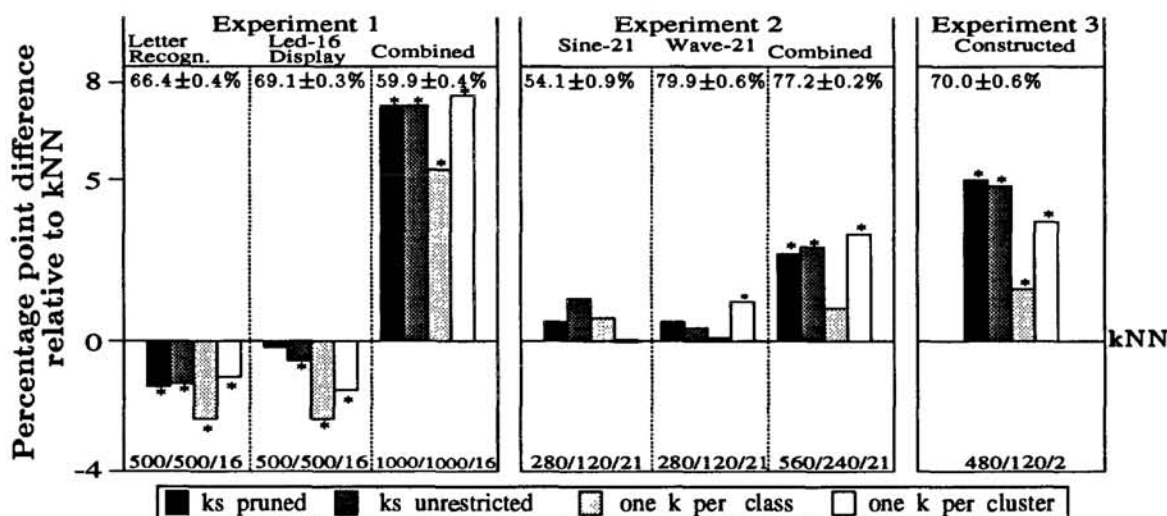

Figure 1: Percent accuracy of local kNN methods relative to kNN on separate test sets. These differences (*) were statistically significant (p < 0.05). Results are based on 25 repetitions. Shown at the bottom of each graph are sizes of training sets/sizes of test sets/number of input features. The percentage at top of each graph indicates average accuracy of kNN ± standard error.

The best performing local methods are localKNN$_{ks\ pruned}$, localKNN$_{ks\ unrestricted}$,

and localKNN$_{one\ k\ per\ cluster}$. These methods were outperformed by kNN in two of the original data sets. However, the performance of these methods was clearly superior to kNN in all domains where data were collections of significantly distinct subsets.

## 3.2   Experiments with Commonly Used Data Sets

Twelve domains of varying sizes and complexities were used to compare the performance of the various nearest neighbor algorithms. Data sets for these domains were obtained from the UC-Irvine repository of machine learning databases (Murphy & Aha, 1991, Aha, 1990, Detrano et al., 1989). Results displayed in Figure 2 indicate that in most data sets which are commonly used to evaluate machine learning algorithms, local nearest neighbor methods have only minor impact on the performance of kNN. The best local methods are either indistinguishable in performance from kNN (localKNN$_{one\ k\ per\ cluster}$) or inferior in only one domain (localKNN$_{ks\ pruned}$).

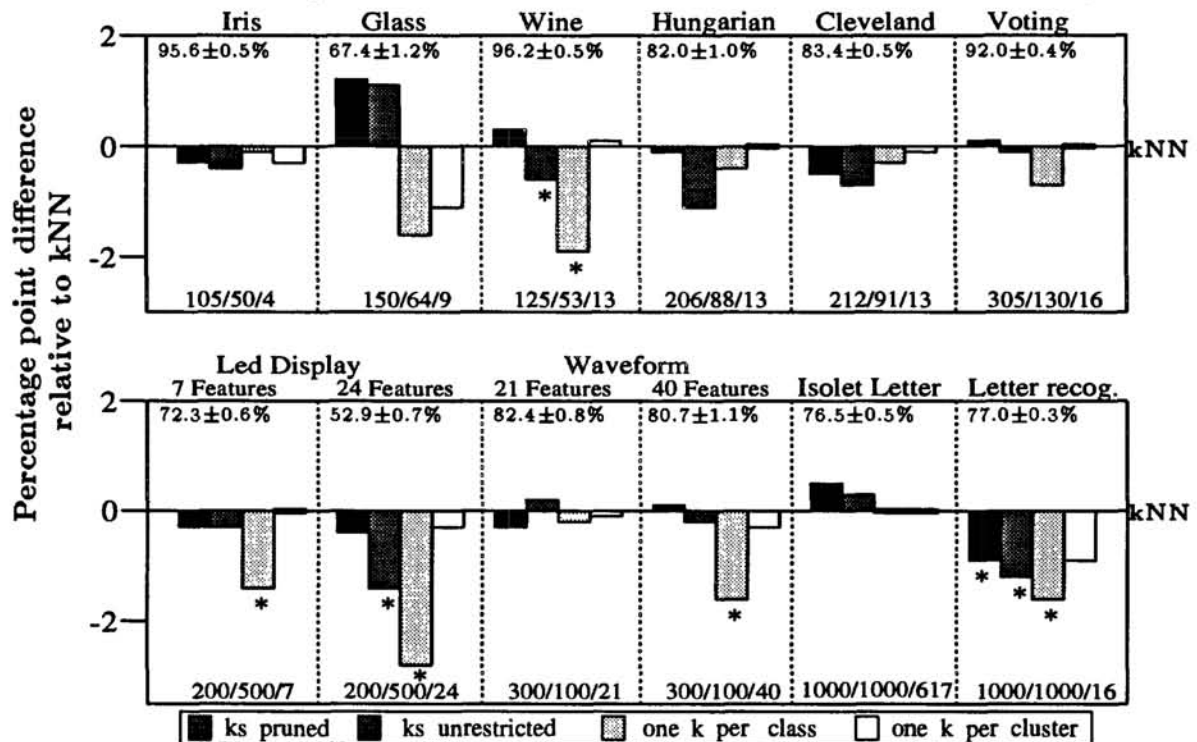

Figure 2: Percent accuracy of local kNN methods relative to kNN on separate test sets. These differences (*) were statistically significant (p < 0.05). Results are based on 25 repetitions. Shown at the bottom of each graph are sizes of training sets/sizes of test sets/number of input features. The percentage at top of each graph indicates average accuracy of kNN ± standard error.

The number of actual $k$ values used varies significantly for the different local methods (Table 1). Not surprisingly, localKNN$_{ks\ unrestricted}$ uses the largest number of distinct $k$ values in all domains. Pruning of $k$s significantly reduced the number of values used in all domains. However, the method using the fewest distinct $k$ values is localKNN$_{one\ k\ per\ cluster}$, which also explains the similar performance of kNN and localKNN$_{one\ k\ per\ cluster}$ in most domains. Note that several clusters computed by localKNN$_{one\ k\ per\ cluster}$ may use the same $k$.

Table 1: Average number of distinct values for $k$ used by local kNN methods.

| Task | kNN | local kNN methods | | | |
|---|---|---|---|---|---|
| | | $ks$ pruned | $ks$ unrestricted | one $k$ per class | one $k$ per cluster |
| Letter recog. | 1 | 7.6±1.1 | 10.8±1.5 | 6.4±0.3 | 1.8±0.2 |
| Led-16 | 1 | 16.4±2.5 | 43.3±0.9 | 9.2±0.1 | 9.2±0.5 |
| Combined$_{LL}$ | 1 | 52.0±3.8 | 71.4±1.2 | 14.7±0.4 | 3.0±0.2 |
| Sine-21 | 1 | 6.6±1.0 | 27.5±1.1 | 2.0±0.0 | 1.0±0.0 |
| Waveform-21 | 1 | 9.1±1.4 | 28.0±1.5 | 2.9±0.1 | 4.2±0.2 |
| Combined$_{SW}$ | 1 | 13.5±1.5 | 30.8±1.6 | 3.0±0.0 | 4.8±0.2 |
| Constructed | 1 | 11.8±0.9 | 15.7±0.5 | 2.0±0.0 | 5.4±0.2 |
| Iris | 1 | 1.6±0.2 | 2.0±0.2 | 2.4±0.1 | 2.3±0.1 |
| Glass | 1 | 7.7±0.8 | 11.2±0.7 | 3.3±0.2 | 1.9±0.2 |
| Wine | 1 | 2.2±0.4 | 3.8±0.4 | 2.0±0.1 | 2.6±0.1 |
| Hungarian | 1 | 4.1±0.6 | 12.6±0.6 | 2.0±0.0 | 1.0±0.0 |
| Cleveland | 1 | 8.0±1.0 | 17.2±1.1 | 1.8±0.1 | 4.6±0.2 |
| Voting | 1 | 4.1±0.4 | 6.4±0.3 | 2.0±0.0 | 1.3±0.1 |
| Led-7 Display | 1 | 5.6±0.4 | 7.6±0.4 | 6.1±0.2 | 1.0±0.0 |
| Led-24 Display | 1 | 16.0±2.9 | 37.4±1.6 | 9.0±0.2 | 1.6±0.2 |
| Waveform-21 | 1 | 9.7±1.3 | 27.8±1.2 | 3.0±0.0 | 4.3±0.1 |
| Waveform-40 | 1 | 8.4±2.0 | 29.9±1.5 | 3.0±0.0 | 4.8±0.1 |
| Isolet Letter | 1 | 11.5±2.1 | 43.9±0.6 | 16.5±0.5 | 7.1±0.3 |
| Letter recog. | 1 | 9.4±1.9 | 17.0±2.3 | 6.0±0.3 | 2.4±0.2 |

Figure 3 shows, for one single run of Experiment 2 (data sets were combined as described in Figure 1), which $k$ values were actually used by the different local methods. Three clusters of $k$ values can be seen in this graph, one cluster at $k = 1$, one at $k = 7,9,11,12$ and the third at $k = 19,20,21$. It is interesting to note that the second and the third cluster correspond to the $k$ values used by kNN in the separate experiments. Furthermore, kNN did not use $k = 1$ in any of the separate runs. This gives insight into why kNN's performance was inferior to that of the local methods in this experiment: Patterns in the combined data set belong to one of three categories as indicated by the $k$ values used to classify them ($k = 1$, $k \approx 10$, $k \approx 20$). Hence, the performance difference is due to the fact that kNN must estimate at training time which single category will give the best performance while the local methods make that decision at classification time for each query depending on its local neighborhood.

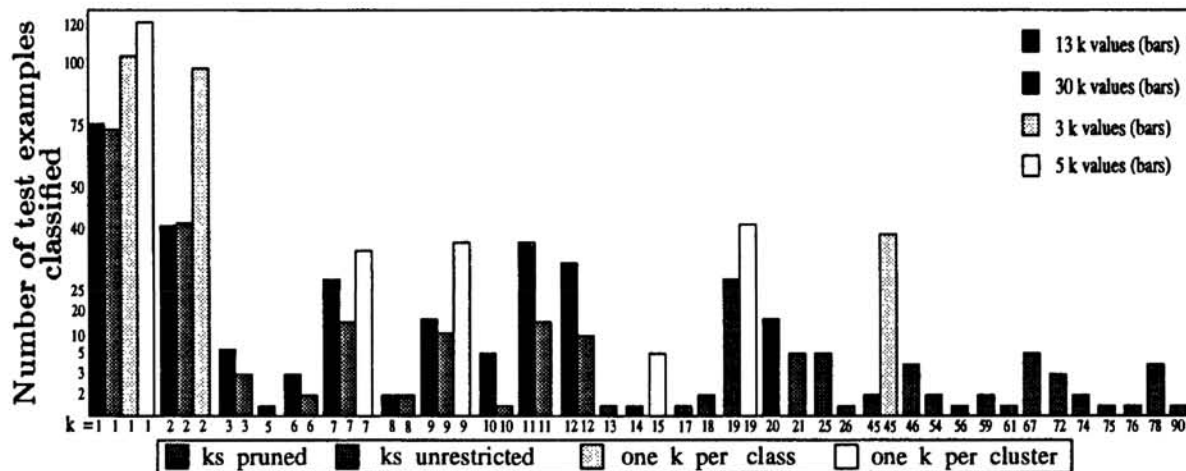

Figure 3: Bars show number of times local kNN methods used certain $k$ values to classify test examples in Experiment 2 (Figure 1 (Combined), numbers based on single run). KNN used $k = 1$ in this experiment.

## 4   Discussion

Four versions of the k-nearest neighbor algorithm which use different values of $k$ for patterns which belong to different regions of the input space were presented and evaluated in this paper. Experiments with constructed and commonly used data sets indicate that local nearest neighbor methods may have superior classification accuracy than kNN in specific domains.

Two methods can be recommended for domains where attributes such as noise or relevance of attributes vary significantly within different parts of the input space. The first method, called localKNN$_{ks\ pruned}$, computes a list of "good" $k$ values for each training pattern, prunes less frequent values from these lists and classifies a query according to the list of $k$ values of a pre-specified number of neighbors of the query. Leave-one-out cross-validation is used to estimate the proper amount of pruning and the size of the neighborhood that should be used.

The other method, localKNN$_{one\ k\ per\ cluster}$, uses a cluster algorithm to determine clusters of input patterns. One $k$ is then computed for each cluster and used to classify queries which fall into this cluster. LocalKNN$_{one\ k\ per\ cluster}$ performs indistinguishable from kNN in all commonly used data sets and outperforms kNN on the constructed data sets. This method compared with all other local methods discussed in this paper introduces a lower computational overhead at classification time and is the only method which could be modified to eliminate the need for leave-one-out cross-validation.

The only purely local method, localKNN$_{ks\ unrestricted}$, performs well on constructed data sets and is comparable to kNN on non-constructed data sets. Sensitivity studies (results not shown) showed that a constant value of 25 for the parameter $M$ gave results comparable to those where cross-validation was used to determine the value of $M$. The advantage of localKNN$_{ks\ unrestricted}$ over the other local methods and kNN is that this method does not require any global information whatsoever (if a constant value for $M$ is used). It is therefore possible to construct a localKNN$_{ks\ unrestricted}$ classifier for each query which makes this method an attractive alternative for on-line learning or extremely large data sets.

If the researcher has reason to believe that the data set used is a collection of subsets with significantly varying attributes such as noise or number of irrelevant features, we recommend the construction of a classifier from the training data using localKNN$_{one\ k\ per\ cluster}$ and comparison of its performance to kNN. If the classifier must be constructed on-line then localKNN$_{ks\ unrestricted}$ should be used instead of kNN.

We conclude that there is considerable evidence that local nearest neighbor methods may significantly outperform the k-nearest neighbor method on specific data sets. We hypothesize that local methods will become relevant in the future when classifiers are constructed that simultaneously solve a variety of tasks.

### Acknowledgements

This research was supported in part by NSF Grant IRI-8657316, NASA Ames Grant NAG 2-630, and gifts from Sun Microsystems and Hewlett-Packard. Many thanks

to Kathy Astrahantseff and Bill Langford for helpful comments during the revision of this manuscript.

## Footnotes

[1]Rival Penalized Competitive Learning is a straightforward modification of the well known k-means clustering algorithm. RPCL's main advantage over k-means clustering is that one can simply initialize it with a sufficiently large number of clusters. Cluster centers are initialized outside of the input range covered by the training examples. The algorithm then moves only those cluster centers which are needed into the range of input values and therefore effectively eliminates the need for cross-validation on the number of clusters in k-means. This paper employed a simple version with the number of initial clusters always set to 25, $\alpha_c$ set to 0.05 and $\alpha_r$ to 0.002.

## References

Aha, D.W. (1990). *A Study of Instance-Based Algorithms for Supervised Learning Tasks.* Technical Report, University of California, Irvine.

Bottou, L., Vapnik, V. (1992). *Local Learning Algorithms.* Neural Computation, 4(6), 888–900.

Dasarathy, B.V. (1991). *Nearest Neighbor(NN) Norms: NN Pattern Classification Techniques.* IEEE Computer Society Press.

Detrano, R., Janosi, A., Steinbrunn, W., Pfisterer, M., Schmid, K., Sandhu, S., Guppy, K., Lee, S. & Froelicher, V. (1989). *Rapid searches for complex patterns in biological molecules.* American Journal of Cardiology, 64, 304–310.

Murphy, P.M. & Aha, D.W. (1991). *UCI Repository of machine learning databases [Machine-readable data repository].* Technical Report, University of California, Irvine.

Weiss, S.M., & Kulikowski, C.A. (1991). *Computer Systems that learn.* San Mateo California: Morgan Kaufmann Publishers, INC.

Xu, L., Krzyzak, A., & Oja, E. (1993). *Rival Penalized Competitive Learning for Clustering Analysis, RBF Net, and Curve Detection* IEEE Transactions on Neural Networks, 4(4),636–649.

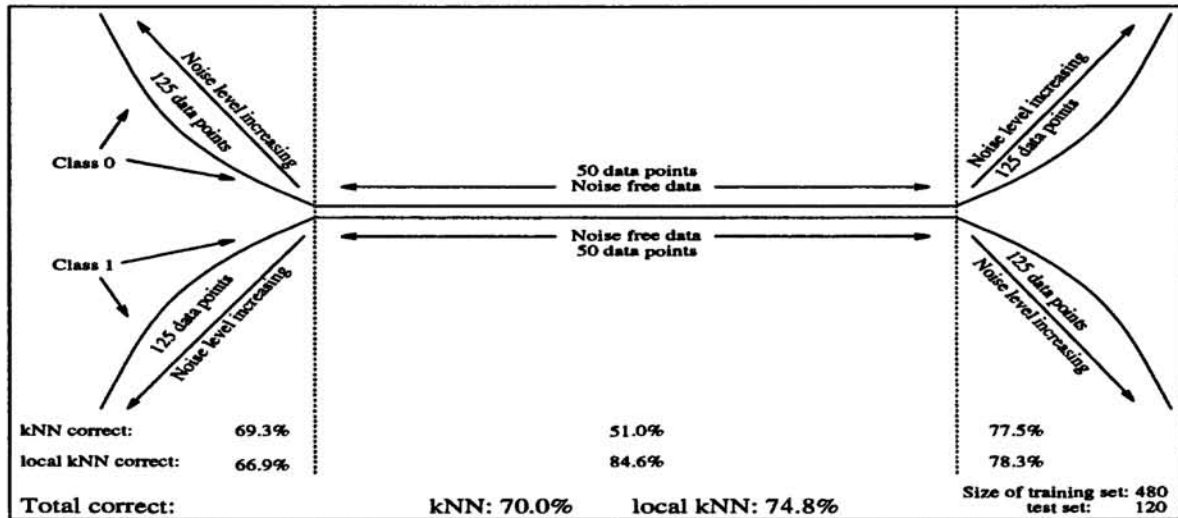

Figure 4: Data points for the Constructed data set were drawn from either of the two displayed curves (i.e. all data points lie on either of the two curves). Class labels were flipped with increasing probabilities to a maximum noise level of approximately 45% at the respective ends of the two lines. Listed at the bottom is performance of kNN and localKNN$_{unrestricted}$ within different regions of the input space and for the entire input space.